# Splines, Rational Functions and Neural Networks

**Robert C. Williamson**
Department of Systems Engineering
Australian National University
Canberra, 2601
Australia

**Peter L. Bartlett**
Department of Electrical Engineering
University of Queensland
Queensland, 4072
Australia

## Abstract

Connections between spline approximation, approximation with rational functions, and feedforward neural networks are studied. The potential improvement in the degree of approximation in going from single to two hidden layer networks is examined. Some results of Birman and Solomjak regarding the degree of approximation achievable when knot positions are chosen on the basis of the probability distribution of examples rather than the function values are extended.

## 1  INTRODUCTION

Feedforward neural networks have been proposed as parametrized representations suitable for nonlinear regression. Their approximation theoretic properties are still not well understood. This paper shows some connections with the more widely known methods of spline and rational approximation. A result due to Vitushkin is applied to determine the relative improvement in degree of approximation possible by having more than one hidden layer. Furthermore, an approximation result relevant to statistical regression originally due to Birman and Solomjak for Sobolev space approximation is extended to more general Besov spaces. The two main results are theorems 3.1 and 4.2.

## 2  SPLINES AND RATIONAL FUNCTIONS

The two most widely studied nonlinear approximation methods are splines with free knots and rational functions. It is natural to ask what connection, if any, these have with neural networks. It is already known that splines with free knots and rational functions are closely related, as Petrushev and Popov's remarkable result shows:

**Theorem 2.1 ([10, chapter 8])** *Let*

$$R_n(f)_p := \inf\{\|f - r\|_p : r \text{ a rational function of degree } n\}$$

$$S_n^k(f)_p := \inf\{\|f - s\|_p : s \text{ a spline of degree } k - 1 \text{ with } n - 1 \text{ free knots}\}.$$

*If $f \in L_p[a,b], \infty < a < b < \infty, 1 < p < \infty, k \geq 1, 0 < \alpha < k$, then*

$$R_n(f)_p = O(n^{-\alpha}) \quad \text{if and only if} \quad S_n^k(f)_p = O(n^{-\alpha}).$$

In both cases the efficacy of the methods can be understood in terms of their flexibility in partitioning the domain of definition: the partitioning amounts to a "balancing" of the error of local linear approximation [4].

There is an obvious connection between single hidden layer neural networks and splines. For example, replacing the sigmoid $(1 + e^{-x})^{-1}$ by the piecewise linear function $(|x + 1| - |x - 1|)/2$ results in networks that are in one dimension splines, and in $d$ dimensions can be written in "Canonical Piecewise Linear" form [3]:

$$f(x) := a + b^T x + \sum_{i=1}^{\kappa} c_i |\alpha_i^T x - \beta_i|$$

defines $f \colon \mathbb{R}^d \to \mathbb{R}$, where $a, c_i, \beta_i \in \mathbb{R}$ and $b, \alpha_i \in \mathbb{R}^d$. Note that canonical piecewise linear representations are unique on a compact domain if we use the form $f(x) := \sum_{i=1}^{\kappa+1} c_i |\alpha_i^T x - 1|$. Multilayer piecewise linear nets are not generally canonical piecewise linear: Let $g(x) := |x+y-1| - |x+y+1| - |x-y+1| - |x-y-1| + x + y$. Then $g(\cdot)$ is canonical piecewise linear, but $|g(x)|$ (a simple two-hidden layer network) is not.

The connection between certain single hidden layer networks and rational functions has been exploited in [13].

## 3  COMPOSITIONS OF RATIONAL FUNCTIONS

There has been little effort in the nonlinear approximation literature in understanding nonlinearly parametrized approximation classes "more complex" than splines or rational functions. Multiple hidden layer neural networks are in this more complex class. As a first step to understanding the utility of these representations we now consider the degree of approximation of certain smooth function classes via rational functions or compositions of rational functions in the sup-metric. A function $\phi \colon \mathbb{R} \to \mathbb{R}$ is rational of degree $\pi$ if $\phi$ can be expressed as a ratio of polynomials in $x \in \mathbb{R}$ of degree at most $\pi$. Thus

$$(3.1) \qquad \phi_\theta := \phi_\theta(x) := \frac{\sum_{i=1}^{\pi} \alpha_i x^i}{\sum_{i=1}^{\pi} \beta_i x^i} \qquad x \in \mathbb{R}, \ \theta := [\alpha, \beta]$$

Let $\sigma_\pi(f,\phi) := \inf\{\|f - \phi_\theta\| : \deg\phi \leq \pi\}$ denote the degree of approximation of $f$ by a rational function of degree $\pi$ or less. Let $\psi := \phi \circ \rho$, where $\phi$ and $\rho$ are rational functions: $\rho: \mathbb{R} \times \Theta_\rho \to \mathbb{R}$, $\phi: \mathbb{R} \times \Theta_\phi \to \mathbb{R}$, both of degree $\pi$. Let $\mathbb{F}$ be some function space (metrized by $\|\cdot\|_\infty$) and let $\sigma_\pi(\mathbb{F}, \cdot) := \sup\{\sigma_\pi(f, \cdot) : f \in \mathbb{F}\}$ denote the degree of approximation of the function class $\mathbb{F}$.

**Theorem 3.1** *Let $\mathbb{F}_\alpha := W_\infty^\alpha(\Omega)$ denote the Sobolev space of functions from a compact subset $\Omega \subset \mathbb{R}$ to $\mathbb{R}$ with $s := \lfloor\alpha\rfloor$ continuous derivatives and the sth derivative satisfying a Lipschitz condition with order $\alpha - s$. Then there exist positive constants $c_1$ and $c_2$ not depending on $\pi$ such that for sufficiently large $\pi$*

$$(3.2) \qquad \sigma_\pi(\mathbb{F}_\alpha, \rho) \geq c_1 \left(\frac{1}{2\pi}\right)^\alpha$$

*and*

$$(3.3) \qquad \sigma_\pi(\mathbb{F}_\alpha, \psi) \geq c_2 \left(\frac{1}{4\pi \log(\pi+1)}\right)^\alpha$$

Note that (3.2) is tight: it is achievable. Whether (3.3) is achievable is unknown. The proof is a consequence of theorem 3.4. The above result, although only for rational functions of a single variable, suggests that no great benefit in terms of degree of approximation is to be obtained by using multiple hidden layer networks.

### 3.1   PROOF OF THEOREM

**Definition 3.2** *Let $\Gamma^d \subset \mathbb{R}^d$. A map $r: \Gamma^d \to \mathbb{R}$ is called a piecewise rational function of degree $k$ with barrier $b_d^q$ of order $q$ if there is a polynomial $b_d^q$ of degree $q$ in $x \in \Gamma^d$ such that on any connected component of $\gamma_i \subset \Gamma^d \setminus \{x : b_d^q(x) = 0\}$, $r$ is a rational function on $\gamma_i$ of degree $k$:*

$$r := r(x) := \frac{P_{d,i}^k(x)}{Q_{d,i}^k(x)} \qquad P_{d,i}^k, Q_{d,i}^k \in \mathbb{R}^d[x].$$

*Note that at any point $x \in \overline{\gamma_i} \cap \overline{\gamma_j}$, $(i \neq j)$, $r$ is not necessarily single valued.*

**Definition 3.3** *Let $\mathbb{F}$ be some function class defined on a set $G$ metrized with $\|\cdot\|_\infty$ and let $\Theta = \mathbb{R}^\nu$. Then $F_{\varepsilon,\nu}^{k,q}: G \times \Theta \to \mathbb{R}$, $F_{\varepsilon,\nu}^{k,q}: (x, \theta) \mapsto F(x, \theta)$ where*

1. *$F(x, \theta)$ is a piecewise rational function of $\theta$ of degree $k$ or less with barrier $b_\nu^{q,x}$ (possibly depending on $x$) of order $q$;*

2. *For all $f \in \mathbb{F}$ there is a $\theta \in \Theta$ such that $\|f - F(\cdot, \theta)\| \leq \varepsilon$;*

*is called an $\varepsilon$-representation of $\mathbb{F}$ of degree $k$ and order $q$.*

**Theorem 3.4 ([12, page 191, theorem 1])** *If $F_{\varepsilon,\nu}^{k,q}$ is an $\varepsilon$-representation of $\mathbb{F}_\alpha$ of degree $k$ and order $q$ with barrier $b$ not depending on $x$, then for sufficiently small $\varepsilon$*

$$(3.4) \qquad \nu \log[(q+1)(k+1)] \geq c \left(\frac{1}{\varepsilon}\right)^{1/\alpha}$$

*where $c$ is a constant not dependent on $\varepsilon, \nu, k$ or $q$.*

Theorem 3.4 holds for any $\varepsilon$-representation $F$ and therefore (by rearrangement of (3.4) and setting $\nu = 2\pi$)

$$(3.5) \qquad \sigma_\pi(\mathbb{F}, F) \geq c \frac{1}{(2\pi \log[(q+1)(k+1)])^\alpha}$$

Now $\phi_\theta$ given by (3.1) is, for any given and fixed $x \in \mathbb{R}$, a piecewise rational function of $\theta$ of degree 1 with barrier of degree 0 (no barrier is actually required). Thus (3.5) immediately gives (3.2).

Now consider $\psi_\theta = \phi \circ \rho$, where

$$\phi = \frac{\sum_{i=1}^{\pi_\phi} \alpha_i y^i}{\sum_{i=1}^{\pi_\phi} \beta_i y^i} \ (y \in \mathbb{R}) \quad \text{and} \quad \rho = \frac{\sum_{j=1}^{\pi_\rho} \gamma_j x^j}{\sum_{j=1}^{\pi_\rho} \delta_j x^j} \ (x \in \mathbb{R}).$$

Direct substitution and rearrangement gives

$$\psi_\theta = \frac{\sum_{i=1}^{\pi_\phi} \alpha_i \left[\sum_{j=1}^{\pi_\rho} \gamma_j x^j\right]^i \left[\sum_{j=1}^{\pi_\rho} \delta_j x^j\right]^{\pi_\phi - i}}{\sum_{i=1}^{\pi_\phi} \beta_i \left[\sum_{j=1}^{\pi_\rho} \gamma_j x^j\right]^i \left[\sum_{j=1}^{\pi_\rho} \delta_j x^j\right]^{\pi_\phi - i}}$$

where we write $\theta = [\alpha, \beta, \gamma, \delta]$ and for simplicity set $\pi_\phi = \pi_\rho = \pi$. Thus $\dim \theta = 4\pi =: \nu$. For arbitrary but fixed $x$, $\psi$ is a rational function of degree $k = \pi$. No barrier is needed so $q = 0$ and hence by (3.4),

$$\sigma_\pi(\mathbb{F}_\alpha, \psi) \geq c_2 \left(\frac{1}{4\pi \log(\pi + 1)}\right)^\alpha.$$

## 3.2 OPEN PROBLEMS

An obvious further question is whether results as in the previous section hold for multivariable approximation, perhaps for multivariable rational approximation.

A popular method of $d$-dimensional nonlinear spline approximation uses dyadic splines [2, 5, 8]. They are piecewise polynomial representations where the partition used is a dyadic decomposition. Given that such a partition $\Xi$ is a subset of a partition generated by the zero level set of a barrier polynomial of degree $\leq |\Xi|$, can Vitushkin's results be applied to this situation? Note that in Vitushkin's theory it is the *parametrization* that is piecewise rational (PR), not the *representation*. What connections are there in general (if any) between PR representations and PR parametrizations?

## 4  DEGREE OF APPROXIMATION AND LEARNING

Determining the degree of approximation for given parametrized function classes is not only of curiosity value. It is now well understood that the statistical sample complexity of learning depends on the size of the approximating class. Ideally the approximat*ing* class is small whilst well approximating as large as possible an approximat*ed* class. Furthermore, in order to make statements such as in [1] regarding the overall degree of approximation achieved by statistical learning, the classical degree of approximation is required.

For regression purposes the metric used is $L_{p,\mu}$, where

$$\|f - g\|_{L_{p,\mu}} := \left[ \int (f(x) - g(x))^p \, d\mu(x) \right]^{1/p}$$

where $\mu$ is a probability measure. Ideally one would like to avoid calculating the degree of approximation for an endless series of different function spaces. Fortunately, for the case of spline approximation (with free knots) this not necessary because (thanks to Petrushev and others) there now exist both direct and converse theorems characterizing such approximation classes. Let $S_n(f)_p$ denote the error of $n$ knot spline approximation in $L_p[0,1]$. Let $I$ denote the identity operator and $T(h)$ the translation operator $(T(h)(f, x) := f(x + h))$ and let $\Delta_h^k := (T(h) - I)^k$, $k = 1, 2, \ldots$, be the difference operators. The modulus of smoothness of order $k$ for $f \in L_p(\Omega)$ is

$$\omega_k(f, t)_p := \sum_{|h| \leq t} \|\Delta_h^k f(\cdot)\|_{L_p(\Omega)}.$$

Petrushev [9] has obtained

**Theorem 4.1** *Let $\tau = (\alpha/d + 1/p)^{-1}$. Then*

$$\text{(4.1)} \qquad \sum_{n=1}^{\infty} [n^\alpha S_n(f)_p]^k \frac{1}{n} < \infty$$

*if and only if*

$$\text{(4.2)} \qquad \int_0^\infty [t^{-\alpha} \omega_k(f, t)_\tau]^\tau \frac{dt}{t}.$$

The somewhat strange quantity in (4.2) is the norm of $f$ in a Besov space $B_{\tau, \tau; k}^\alpha$. Note that for $\alpha$ large enough, $\tau < 1$. That is, the smoothness is measured in an $L_p$ $(p < 1)$ space. More generally [11], we have (on domain $[0, 1]$)

$$\|f\|_{B_{p,q;k}^\alpha} := \left( \int_0^1 (t^{-\alpha} \omega_k(f, t)_p)^q \frac{dt}{t} \right)^{1/q}$$

Besov spaces are generalizations of classical smoothness spaces such as Sobolev spaces (see [11]).

We are interested in approximation in $L_{p,\mu}$ and following Birman and Solomjak [2] ask what degree of approximation in $L_{p,\mu}$ can be obtained when the knot positions are chosen according to $\mu$ rather than $f$. This is of interest because it makes the problem of determining the parameter values on the basis of observations linear.

**Theorem 4.2** *Let $f \in L_{p,\mu}$ where $\mu \in L_\lambda$ for some $\lambda > 1$ and is absolutely continuous. Choose the $n$ knot positions of a spline approximant $v$ to $f$ on the basis of $\mu$ only. Then for all such $f$ there is a constant $c$ not dependent on $n$ such that*

$$\text{(4.3)} \qquad \|f - v\|_{L_{p,\mu}} \leq c n^{-\alpha} \|f\|_{B_{\sigma, \sigma; k}^\alpha}$$

*where $\sigma = (\alpha + (1 - \lambda^{-1})p^{-1})^{-1}$ and $p < \sigma$. The constant $c$ depends on $\mu$ and $\lambda$.*

*If $p \geq 1$ and $\sigma \leq p$, for any $\alpha < \sigma^{-1}$ for all $f$ under the conditions above, there is a $v$ such that*

(4.4)
$$||f - v||_{L_{p,\mu}} \leq cn^{-\alpha + \frac{1}{\sigma} - \frac{1}{p}}||f||_{B_{\sigma;k}^\alpha}$$

*and again $c$ depends on $\mu$ and $\lambda$ but does not depend on $n$.*

**Proof**  First we prove (4.3). Let $[0,1]$ be partitioned by $\Xi$. Thus if $v$ is the approximant to $f$ on $[0,1]$ we have

$$||f - v||_{L_{p,\mu}}^p = \sum_{\Delta \in \Xi} ||f - v||_{L_{p,\mu}(\Delta)}^p = \sum_{\Delta \in \Xi} \int_\Delta |f(x) - v(x)|^p d\mu(x).$$

For any $\lambda \geq 1$,

$$\int_\Delta |f(x) - v(x)|^p d\mu(x) = \int_\Delta |f - v|^p \left(\frac{d\mu}{dx}\right) dx$$

$$\leq \left[\int_\Delta |f - v|^{p(1-\lambda^{-1})^{-1}} dx\right]^{1-\lambda^{-1}} \left[\int_\Delta \left(\frac{d\mu}{dx}\right)^\lambda dx\right]^{\lambda^{-1}}$$

$$= ||f - v||_{L_\psi(\Delta)}^p ||d\mu/dx||_{L_\lambda(\Delta)}$$

where $\psi = p(1 - \lambda^{-1})^{-1}$. Now Petrushev and Popov [10, p.216] have shown that there exists a polynomial of degree $k$ on $\Delta = [r, s]$ such that

$$||f - v||_{L_\psi(\Delta)}^p \leq c||f||_{B(\Delta)}^p$$

where

$$||f||_{B(\Delta)} := \left(\int_0^{(s-r)/k} (t^{-\alpha}||\Delta_t^k f(\cdot)||_{L_\sigma(r,s-kt)})^\sigma \frac{dt}{t}\right)^{1/\sigma}$$

and $\sigma := (\alpha + \psi^{-1})^{-1}$, $0 < \psi < \infty$ and $k \geq 1$. Let $|\Xi| =: n$ and choose $\Xi = \cup_i \Delta_i$ ($\Delta_i = [r_i, s_i]$) such that

$$\int_{\Delta_i} \left(\frac{d\mu}{dx}\right)^\lambda dx = \frac{1}{n}||d\mu/dx||_{L_\lambda(0,1)}^\lambda.$$

Thus $||d\mu/dx||_{L_\lambda(\Delta)} = n^{-1/\lambda}||d\mu/dx||_{L_\lambda(0,1)}$. Hence

(4.5)
$$||f - v||_{L_{p,\mu}}^p \leq c||d\mu/dx||_{L_\lambda} \sum_{\Delta \in \Xi} n^{-1/\lambda}||f||_{B(\Delta)}^p.$$

Since (by hypothesis) $p < \sigma$, Holder's inequality gives

$$||f - v||_{L_{p,\mu}}^p \leq c||d\mu/dx||_{L_\lambda} \left[\sum_{\Delta \in \Xi} \left(\frac{1}{n}\right)^{\frac{1}{\lambda} \frac{1}{1-p/\sigma}}\right]^{1-\frac{p}{\sigma}} \left[\sum_{\Delta \in \Xi} ||f||_{B(\Delta)}^\sigma\right]^{\frac{p}{\sigma}}$$

Now for arbitrary partitions $\Xi$ of $[0,1]$ Petrushev and Popov [10, page 216] have shown

$$\sum_{\Delta \in \Xi} ||f||_{B(\Delta)}^\sigma \leq ||f||_{B_{\sigma;k}^\alpha}^\sigma$$

where $B^\alpha_{\sigma;k} = B^\alpha_{\sigma,\sigma;k} = B([0,1])$. Hence

$$||f - v||^p_{L_{p,\mu}} \le c||d\mu/dx||_{L_\lambda}\ n^{\frac{p}{\sigma}+1-\frac{1}{\lambda}}\ ||f||^p_{B^\alpha_{\sigma;k}}$$

and so

$$(4.6) \qquad ||f - v||_{L_{p,\mu}} \le c||d\mu/dx||^{1/p}_{L_\lambda}\ n^{-\alpha}\ ||f||_{B^\alpha_{\sigma;k}}$$

with $\sigma = (\alpha + \psi^{-1})^{-1}$, $\psi = p(1 - \lambda^{-1})^{-1}$. Hence $\sigma = (\alpha + \frac{1-\lambda^{-1}}{p})^{-1}$. Thus given $\alpha$ and $p$, choosing different $\lambda$ adjusts the $\sigma$ used to measure $f$ on the right-hand side of (4.6). This proves (4.3).

Note that because of the restriction that $p < \sigma$, $\alpha > 1$ is only achievable for $p < 1$ (which is rarely used in statistical regression [6]). Note also the effect of the term $||d\mu/dx||^{1/p}_{L_\lambda}$. When $\lambda = 1$ this is identically 1 (since $\mu$ is a probability measure). When $\lambda > 1$ it measures the departure from uniform distribution, suggesting the degree of approximation achievable under non-uniform distributions is worse than under uniform distributions.

Equation (4.4) is proved similarly. When $\sigma \le p$ with $p \ge 1$, for any $\alpha \le 1/\sigma$, we can set $\lambda := (1 - \frac{p}{\sigma} + p\alpha)^{-1} \ge 1$. From (4.5) we have

$$||f - v||^p_{L_{p,\mu}} \le c||d\mu/dx||_{L_\lambda} \sum_{\Delta \in \Xi} \left(\frac{1}{n}\right)^{1/\lambda} ||f||^p_{B(\Delta)}$$

$$\le c||d\mu/dx||_{L_\lambda} \left(\frac{1}{n}\right)^{1/\lambda} \left[\sum_{\Delta \in \Xi} ||f||^\sigma_{B(\Delta)}\right]^{p/\sigma}$$

$$\le c||d\mu/dx||_{L_\lambda} n^{-1+\frac{p}{\sigma}-p\alpha} ||f||^p_{B^\alpha_{\sigma;k}}$$

and therefore

$$||f - v||_{L_{p,\mu}} \le c||d\mu/dx||^{1/p}_{L_\lambda} n^{-\alpha+\frac{1}{\sigma}-\frac{1}{p}} ||f||_{B^\alpha_{\sigma;k}}.$$

∎

## 5   CONCLUSIONS AND FURTHER WORK

In this paper a result of Vitushkin has been applied to "multi-layer" rational approximation. Furthermore, the degree of approximation achievable by spline approximation with free knots when the knots are chosen according to a probability distribution has been examined.

The degree of approximation of neural networks, particularly multiple layer networks, is an interesting open problem. Ideally one would like both direct and converse theorems, completely characterizing the degree of approximation. If it turns out that from an approximation point of view neural networks are no better than dyadic splines (say), then there is a strong incentive to study the PAC-like learning theory (of the style of [7]) for such spline representations. We are currently working on this topic.

## Acknowledgements

This work was supported in part by the Australian Telecommunications and Electronics Research Board and OTC. The first author thanks Federico Girosi for providing him with a copy of [4]. The second author was supported by an Australian Postgraduate Research Award.

## References

[1]  A. R. Barron, Approximation and Estimation Bounds for Artificial Neural Networks, To appear in Machine Learning, 1992.

[2]  M. S. Birman and M. Z. Solomjak, Piecewise-Polynomial Approximations of Functions of the Classes $W_p^\alpha$, *Mathematics of the USSR — Sbornik*, **2** (1967), pp. 295–317.

[3]  L. Chua and A. -C. Deng, Canonical Piecewise-Linear Representation, *IEEE Transactions on Circuits and Systems*, **35** (1988), pp. 101–111.

[4]  R. A. DeVore, Degree of Nonlinear Approximation, in *Approximation Theory VI, Volume 1*, C. K. Chui, L. L. Schumaker and J. D. Ward, eds., Academic Press, Boston, 1991, pp. 175–201.

[5]  R. A. DeVore, B. Jawerth and V. Popov, Compression of Wavelet Decompositions, To appear in American Journal of Mathematics, 1992.

[6]  H. Ekblom, $L_p$-methods for Robust Regression, *BIT*, **14** (1974), pp. 22–32.

[7]  D. Haussler, Decision Theoretic Generalizations of the PAC Model for Neural Net and Other Learning Applications, Report UCSC-CRL-90-52, Baskin Center for Computer Engineering and Information Sciences, University of California, Santa Cruz, 1990.

[8]  P. Oswald, On the Degree of Nonlinear Spline Approximation in Besov-Sobolev Spaces, *Journal of Approximation Theory*, **61** (1990), pp. 131–157.

[9]  P. P. Petrushev, Direct and Converse Theorems for Spline and Rational Approximation and Besov Spaces, in *Function Spaces and Applications (Lecture Notes in Mathematics 1302)*, M. Cwikel, J. Peetre, Y. Sagher and H. Wallin, eds., Springer-Verlag, Berlin, 1988, pp. 363–377.

[10]  P. P. Petrushev and V. A. Popov, *Rational Approximation of Real Functions*, Cambridge University Press, Cambridge, 1987.

[11]  H. Triebel, *Theory of Function Spaces*, Birkhäuser Verlag, Basel, 1983.

[12]  A. G. Vitushkin, *Theory of the Transmission and Processing of Information*, Pergamon Press, Oxford, 1961, Originally published as *Otsenka slozhnosti zadachi tabulirovaniya* (Estimation of the Complexity of the Tabulation Problem), Fizmatgiz, Moscow, 1959.

[13]  R. C. Williamson and U. Helmke, Existence and Uniqueness Results for Neural Network Approximations, Submitted, 1992.